# Neural Combinatorial Optimization for Robust Routing Problem with Uncertain Travel Times

**Pei Xiao[1], Zizhen Zhang[1*], Jinbiao Chen[1], Jiahai Wang[1], Zhenzhen Zhang[2]**
School of Computer Science and Engineering, Sun Yat-sen University, P.R. China [1]
School of Economics and Management, Tongji University, P.R. China [2]
`xiaop39@mail2.sysu.edu.cn, zhangzzh7@mail.sysu.edu.cn`
`chenjb69@mail2.sysu.edu.cn, wangjiah@mail.sysu.edu.cn`
`zhenzhenzhang@tongji.edu.cn`

## Abstract

We consider the robust routing problem with uncertain travel times under the min-max regret criterion, which represents an extended and robust version of the classic traveling salesman problem (TSP) and vehicle routing problem (VRP). The general budget uncertainty set is employed to capture the uncertainty, which provides the capability to control the conservatism of obtained solutions and covers the commonly used interval uncertainty set as a special case. The goal is to obtain a robust solution that minimizes the maximum deviation from the optimal routing time in the worst-case scenario. Given the significant advancements and broad applications of neural combinatorial optimization methods in recent years, we present our initial attempt to combine neural approaches for solving this problem. We propose a dual multi-head cross attention mechanism to extract problem features represented by the inputted uncertainty sets. To tackle the built-in maximization problem, we derive the regret value by invoking a pre-trained model, subsequently utilizing it as the reward during the model training. Our experimental results on the robust TSP and VRP demonstrate the efficacy of our neural combinatorial optimization method, showcasing its ability to efficiently handle the robust routing problem of various sizes within a shorter time compared with alternative heuristic approaches.

## 1 Introduction

The classic Traveling Salesman Problem (TSP) and Vehicle Routing Problem (VRP) represent fundamental NP-hard combinatorial optimization challenges. In these routing problems, an agent commences from a designated node and fulfills specific task requisites. The primary objective is to minimize the total travel time or cost of access. However, the classic routing problem is mathematically simplified and idealized, lacking the necessary robustness to handle the complexities of the real world. In actual scenarios, the existence of various uncertain factors may significantly impact the final outcome. For example, considering the travel time between two nodes, it is not only affected by objective factors such as traffic conditions, road quality, and weather, but also by subjective factors such as the traveler's status and mood. When planning a trip and estimating travel time, uncertainty about the actual travel conditions may arise. To ensure a satisfactory travel experience, it is sensible to adopt a robust approach, which involves seeking a solution that is less vulnerable to adverse conditions, even in the worst-case scenario.

While there are several robust routing formulations in the literature, this study primarily focuses on classic robust routing problems featuring a *budget uncertainty set*. The underlying idea is that the uncertainty support set is represented by a set of interval data, but usually not all arcs are affected to

achieve their worst values simultaneously in one realistic scenario. More concretely, the travel time between two nodes is given by a range with upper and lower values, and a parameter $\Gamma$ is used to specify the number of affected arcs. In addition, the *min-max regret* criterion (also known as *robust deviation criterion* [1]) is generally considered, which avoids overconservatism while considering certain degree of robustness. It aims to minimize the maximum deviation from the optimal route over the realization of edge costs in all scenarios. Detailed definition of such criterion will be provided in Section 3.

To tackle the robust routing problem, traditional robust optimization methods have been employed, including both exact and heuristic approaches. These methods treat the problem instances individually and aim to generate optimal or near-optimal robust solutions within a reasonable computation time. However, their efficiency diminishes when dealing with a large set of instances that exhibit similar interval uncertainty structures. Specifically, the uncertain travel time of edges in different instances may share comparable upper and lower limits, following certain hourly or daily patterns. To overcome these limitations, we can leverage advanced neural techniques to exploit the underlying patterns and similarities in the data, thus enabling the derivation of more efficient solutions.

In summary, this paper attempts to explore the neural combination optimization methods to address the robust routing problem with uncertain travel times characterized by budget uncertainty set. The contributions of our work can be highlighted as follows.

- We introduce the robust routing problem with the general budget uncertainty set, which covers the interval uncertainty set as a special case, and treat the problem from the perspective of deep reinforcement learning.

- We propose an end-to-end neural model to capture the features of robust routing problem, and use a pre-trained neural model to efficiently calculate the reward with respect to the worst-case scenario during training.

- We conduct extensive experiments on Robust Traveling Salesman Problem (RTSP) and Robust Capacitated Vehicle Routing Problem (RVCRP) instances. The results substantiate the efficacy of our approach in efficiently handling robust routing problems across diverse scales within shorter computation time.

The remainder of the paper is structured as follows. In Section 2, we provide an overview of existing works related to robust optimization and neural combinatorial optimization, respectively. In Section 3, taking the Robust Traveling problem as an example, we describe the problem definition of RTSP in detail. In Section 4, we elaborate the proposed neural combinatorial optimization method. In Section 5, we provide experimental results for different RTSP and RCVRP instances. In Section 6, we conclude our work with possible future directions. [1]

## 2 Related Work

**Robust Optimization.** As early as the 1950s, Bellman and Zadeh [2] initiated research on uncertainty optimization. Robust optimization, as introduced by Ben-Tal et al. [3], adopts a conservative approach based on worst-case optimization, ensuring that the obtained solution remains feasible for any possible realization of uncertain parameters. The formulation of robust optimization models varies significantly depending on the choice of uncertainty sets and robustness standards. Uncertainty sets encompass various types, including interval sets [3], ellipsoid sets [4, 3], polyhedron sets [5], budget sets [6], and more. Robustness standards encompass criteria such as the normal min-max standard, min-max-regret standard [1], adjustable robustness [7], etc. Considering the literature of RTSP, Montemanni et al. [8] proposed exact algorithms for RTSP with interval uncertainty sets, employing the robust deviation criterion. Chassein and Goerigk [9] introduced a recoverable robust model that allows a tour to modify a limited number of edges once a scenario becomes known. Lu et al. [10] proposed a heuristic method based on simulated annealing and the LKH algorithm to solve RTSP with the min-max regret criterion and interval uncertainty sets. Additionally, Hasegawa and Wu [11] presented an intriguing heuristic edge generation algorithm to tackle this problem. On the issue of Robust VRP, Sungur et al. [12] applied the MTZ model to the traditional CVRP problem and

robustly modeled demand uncertainty. Subsequently, Solano-Charris et al. [13] introduced a meta-heuristic algorithm based on local search to address the lexicographic min-max criterion. Eufinger et al. [14] proposed a heuristic algorithm for solving the $k$-adaptability min-max-min criterion.

**Neural Combinatorial Optimization.** Neural combinatorial optimization methods for end-to-end solving of the Vehicle Routing Problem (VRP) can be broadly categorized into supervised learning (SL) and deep reinforcement learning (DRL) approaches [15]. While SL requires high-quality solutions to the VRP as training labels [16–18], DRL methods are divided into value-based [19–21] and policy-based techniques. Given the complexity of routing policies, policy-based DRL approaches have become the mainstream choice, achieving notable success in various combinatorial optimization problems [22–24], including routing [25–28]. In recent years, advancements have continued. For example, Zhang et al. [29] introduced a DRL method employing an edge-based graph attention network for solving the practical Vehicle Routing Problem with Time Windows (VRPTW). Additionally, Kwon et al. [30] designed a neural model utilizing cross attention to handle matrix inputs, while Zhou et al. [31] presented a general meta-learning framework that considers the generalization of size and distribution in vehicle routing problems. These studies collectively contribute to the advancement of neural combinatorial optimization methods, showcasing their ability to tackle various types of routing problems.

**Neural Methods for Robust Optimization.** In recent studies, there have been initial attempts to apply neural methods to the field of robust optimization. Jacobs et al. [32] employed reinforcement learning to solve min-max robust optimization problems. They precisely solved the inner minimization problem and utilized the DQN method to handle the outer problem. Dumouchelle et al. [33] proposed an efficient machine-learning-driven instantiation of the column-and-constraint generation algorithm for two-stage min-max-min robust optimization problems. Our approach differs in that we solely train a neural model capable of rapidly constructing high-quality solutions through inference. By focusing on training a neural model, we aim to provide a novel perspective on addressing the robust routing problem.

## 3 Problem Description

In the subsequent discussion, we employ RTSP as a paradigmatic illustration of the robust routing problem for comprehensive exposition. More detailed formulations of RTSP and RCVRP can be referenced in Appendix 7.3.

### 3.1 Nominal TSP

The traveling salesman problem is a classic combinatorial optimization problem mathematically defined as follows. Given a complete symmetric graph $G = (V, E)$, where $V = \{1, 2..., n\}$ is the node set, $t_{ij}$, as a nominal value, denote the expected travel time of edge $(i, j) \in E$. Here, $x_{ij}$ is a binary variable that indicates whether the edge $(i, j)$ is selected for the route, with 1 for selection and 0 for non-selection. The objective is to find a close route with the minimum total travel time.

For simplicity, we hereinafter use $\mathbb{S}$ to represent the solution space, which is the set of all feasible TSP routes.

### 3.2 Robust TSP

Based on historical data, it is relatively straightforward to determine the fluctuation range $[t_{ij}^-, t_{ij}^+]$ of travel time for each edge $(i, j)$, where $t_{ij}^-$ and $t_{ij}^+$ denote the lower and upper bounds of possible values, respectively. The *interval uncertainty set* is defined as $\mathbb{U}_{interval} = \{t | t_{ij} \in [t_{ij}^-, t_{ij}^+], \forall (i, j) \in E\} = \{t | t_{ij} = t_{ij}^- + \hat{t}_{ij} \eta_{ij}, 0 \leq \eta_{ij} \leq 1, \forall (i, j) \in E\}$, where $\hat{t}_{ij} = t_{ij}^+ - t_{ij}^-$. To provide flexibility to control the conservatism of obtained solutions, the *budget uncertainty set* is usually considered, which is defined as $\mathbb{U}_{budget} = \{t | t_{ij} = t_{ij}^- + \hat{t}_{ij} \eta_{ij}, 0 \leq \eta_{ij} \leq 1, \forall (i, j) \in E; \sum_{(i,j) \in E} \eta_{ij} \leq \Gamma\}$. Here, the parameter $\Gamma$ controls the number of affected edges. It is apparent that the budget uncertainty set simplifies to the interval uncertainty set when $\Gamma \geq \binom{n}{2}$ and to the deterministic version when $\Gamma = 0$. Hence, the budget uncertainty set demonstrates a certain level of generalization.

To specify the objective of RTSP, the *min-max-regret* criterion is adopted. For better understanding, we first introduce the concept of *regret*.

**Definition 3.1** (**regret**). *Given a TSP solution $x$ and a scenario $u$, the* regret *value denotes the difference between the objective value of $x$ and the optimal solution under $u$.*

$$regret(x, u) = \sum_{(i,j) \in E} t_{ij}^u x_{ij} - \min_{y \in \mathbb{S}} \sum_{(i,j) \in E} t_{ij}^u y_{ij}, \tag{1}$$

*where $t_{ij}^u$ is the actual travel time of edge $(i, j)$ under scenario $u$. $y$ is the 0-1 decision variable of nominal TSP under scenario $u$.*

**Definition 3.2** (**min-max-regret criterion**). *It aims to find a TSP solution that minimizes the maximum* regret *over all realizations of edge costs.*

$$\min_{x \in \mathbb{S}} \max_{u \in \mathbb{U}} regret(x, u), \tag{2}$$

*where $\mathbb{U}$ is the uncertainty set including all possible scenarios.*

From Equation (2), it is evident that RTSP is difficult to solve directly due to its triple optimality judgment. However, it is worth mentioning that in Montemanni et al. [8], a theorem regarding the derivation of the worst-case scenario in budget uncertainty set is proposed under the min-max-regret criterion. By leveraging this theorem, the complexity of RTSP can be significantly reduced.

**Theorem 1.** *Given a TSP solution $x$, the worst-case scenario in budget uncertainty set for solution $x$ (i.e., the scenario that maximizes the regret of solution $x$) is the one where exactly $\Gamma$ edges with the largest upper bound in $x$ takes its upper bound (i.e., $\eta_{ij} = 1$ and $t_{ij} = t_{ij}^- + \hat{t}_{ij}$), and the remaining edges take its lower bound (i.e., $\eta_{ij} = 0$ and $t_{ij} = t_{ij}^-$).*

*Proof.* See Appendix 7.1. □

Given that the interval uncertainty set is a special case of the budget uncertainty set with $\Gamma \geq \binom{n}{2}$, a corollary can be outlined as follows.

**Corollary 1.** *Given a TSP solution $x$, the worst-case scenario in interval uncertainty set for solution $x$ (i.e., the scenario that maximizes the regret of solution $x$) is the one where all the traversed edges of solution $x$ reach their upper bound limit and the remaining edges reach their lower bound limit.*

Taking the interval uncertainty for example, let $wc(x)$ denote the worst-case scenario for solution $x$. Its associated travel time $t_{ij}^{wc(x)}$ is given by:

$$t_{ij}^{wc(x)} = x_{ij} t_{ij}^+ + (1 - x_{ij}) t_{ij}^-, \quad \forall (i, j) \in E \tag{3}$$

Therefore, the formulation of RTSP can be simply expressed as:

$$\min_{x \in \mathbb{S}} \{ \sum_{(i,j) \in E} t_{ij}^+ x_{ij} - \min_{y \in \mathbb{S}} \sum_{(i,j) \in E} t_{ij}^{wc(x)} y_{ij} \}. \tag{4}$$

In Figure 1, we give a specific example to illustrate the solution process of RTSP with interval uncertainty. There are 4 nodes and 6 edges. Each edge is associated with an uncertainty interval (see Figure 1a). Assuming that a feasible solution $x$ is "$1 \rightarrow 2 \rightarrow 3 \rightarrow 4 \rightarrow 1$" represented by the green edges, its nominal total time is $5 + 9 + 4 + 4 = 22$. According to Corollary 1, we can obtain the worst-case scenario $wc(x)$ with respect to $x$, as indicated by the edge values in Figure 1b. In this case, we can determine the optimal solution under the scenario $wc(x)$, represented by the red edges in Figure 1c, with a total time $5 + 1 + 4 + 4 = 14$. The max-regret value for $x$ is then calculated as $22 - 14 = 8$. By obtaining the *max-regret* for every feasible solution, the final optimal solution is the one with the minimum *max-regret*.

## 4   Methodology

The objective of the robust routing problem is to discover a robust route that minimizes the *max-regret*. Only when a route is complete can we calculate its *max-regret* value. This observation bears resemblance to the concept of delayed reward in deep reinforcement learning. We propose employing the neural combinatorial optimization method to construct such route in an end-to-end manner. To elaborate on our approach, we also employ RTSP with interval uncertainty for the study.

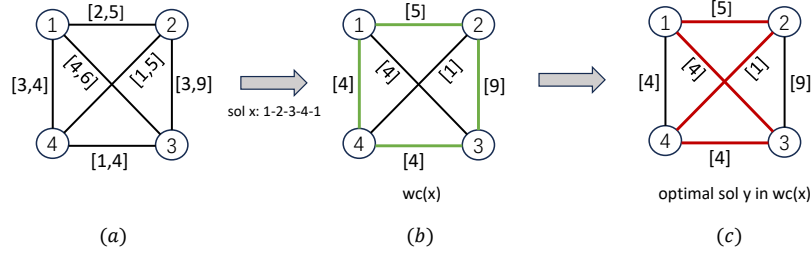

(a)　(b)　(c)

Figure 1: An instance of RTSP with interval uncertainty. (a) Interval support set. (b) Worst case scenario $wc(x)$ corresponding to route $x$. (c) Optimal TSP solution under scenario $wc(x)$.

## 4.1 Solution Framework

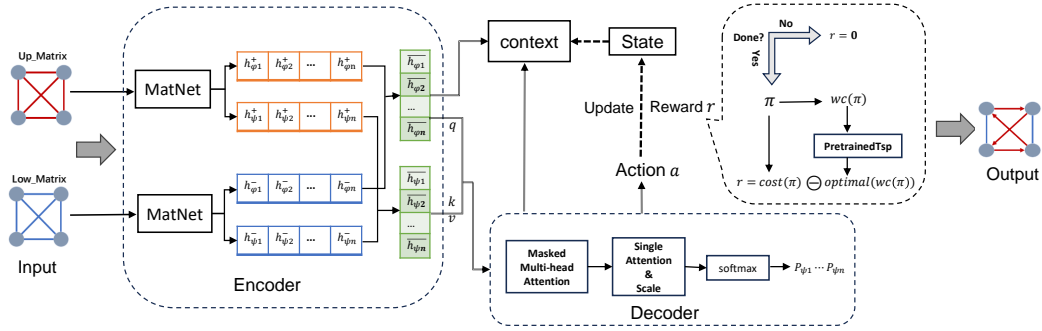

Figure 2: The complete solution framework of our method. Variable $a$ represents the action of selecting the next node according to the probability distribution. The reward $r$ is a sparse variable that takes its max-regret value only in the complete sequence, while being set to zero for all other time steps.

The overall solution framework, as illustrated in Figure 2, is based on the encoder-decoder model. In this model, the encoder component receives the input data and exploits a series of neural networks to extract embeddings that represent the problem features. These embeddings are then utilized by the decoder component to compute probabilities for node selection and construct a complete solution in an autoregressive manner. Specifically, the model trains a stochastic policy $p(\pi|s)$ to generate a solution $x$, or a permutation $\pi$ equivalently, for an instance $s$. It is parameterized by $\theta$ as:

$$p_\theta(\pi|s) = \prod_{t=1}^{n} p_\theta(\pi_t|s, \pi_{1:t-1}). \tag{5}$$

Note that the input data consists of a dual-weighted graph, where each edge is associated with an upper value and a lower value. Taking inspiration from MatNet [30], we treat the input as two matrices: one composed of the upper range values ("Up_Matrix"), and the other composed of the lower range values ("Low_Matrix"). These matrices are then fed into the encoder.

Furthermore, the model requires a reward signal for adjusting the network weight parameters. Once all the decoding steps are complete, we obtain a nominal solution $\pi$ for the instance $s$. The loss function is defined as:

$$L(\theta|s) = \mathbb{E}_{p_\theta(\pi|s)}[R(\pi)], \tag{6}$$

where $R(\pi)$ is the expectation of the robustness of $\pi$, or its (negative) reward, given by

$$R(\pi) = t^+(\pi) - TSP^*(wc(\pi)). \tag{7}$$

Equation (7) is originated from Equation (4), where $t^+(\pi)$ is the total travel time of solution $\pi$ using the upper range value. $TSP^*(wc(\pi))$ calculates the optimal TSP route under the worst-case scenario

with respect to solution $\pi$. Because a large number of samples has to be generated to train the model, we need to find a solver to tackle TSP efficiently. In fact, it is not necessary to exactly solve TSP in the training stage due to the expectation of $R(\pi)$ and high complexity of TSP. We turn to approximate optimal solutions by using a pre-trained TSP model detailed in Section 4.3.

## 4.2 Model Architecture

### 4.2.1 Encoder

The encoder takes two matrices ("Up_Matrix" and "Low_Matrix") as the input. Each matrix can be regarded as a bipartite graph $G = (\Phi, \Psi, D)$, where $\Phi = (\phi_1, \ldots, \phi_n)$ is the set of nodes with outgoing edges, $\Psi = (\psi_1, \ldots, \psi_n)$ is the set of nodes with incoming edges, and $D = [d_{ij}]_{n \times n}$ gives the edge values by the corresponding matrix.

We extract the feature embeddings of $\Phi$ and $\Psi$, denoted as $h_\Phi$ and $h_\Psi$, using MatNet [30] (detailed architecture can be found in Appendix 7.7). To achieve this, we initialize the initial embedding $h^0_{\phi_i}$ for each $\phi_i \in \Phi$ as a zero vector, enabling the model to support variable-sized inputs. For each $\psi_j \in \Psi$, the initial embedding $h^0_{\psi_j}$ is initialized as a one-hot vector to obtain discriminative first attention layer representations. Next, these initial embeddings are combined with the corresponding matrix $D$ and passed through $L$ attention layers to iteratively update the embeddings. It should be noted that the encoding process for sets $\Phi$ and $\Psi$ is performed mutually. Specifically, for the encoding process of set $\Phi$, $h_\Phi$ serves as the query vector, $h_\psi$ acts as the key-value vector, and the matrix $D$ is utilized for weighted fusion. Therefore, we obtain the updated embedding $h'_\Phi$ as the output of the MatNet: $h'_\Phi = \text{MatNet}(h_\Phi, h_\Psi, D)$. Conversely, for the encoding process of set $\Psi$, we swap the roles of $h_\Phi$ and $h_\Psi$ and employ the transpose of matrix $D$. Consequently, we have $h'_\Psi = \text{MatNet}(h_\Psi, h_\Phi, D^T)$.

Since there are two matrices involved, we use $h^+_\Phi$ and $h^+_\Psi$ to denote the output embeddings for "Up_Matrix". Similarly, we use $h^-_\Phi$ and $h^-_\Psi$ to denote the output embeddings for "Low_Matrix". Finally, the graph embeddings, denoted as a pair $(\overline{h_\Phi}, \overline{h_\Psi})$, are obtained by fusing the features of "Up_Matrix" and "Low_Matrix". We simply perform the summation on their embeddings: $\overline{h_\Phi} = h^+_\Phi + h^-_\Phi, \overline{h_\Psi} = h^+_\Psi + h^-_\Psi$.

### 4.2.2 Decoder

The decoder module generates a complete solution in an autoregressive manner based on the graph embeddings $(\overline{h_\Phi}, \overline{h_\Psi})$ obtained from the encoder.

The key component of the decoder is the *masked multi-head cross attention* mechanism. At each decoding step $t$, it utilizes $\overline{h_{\psi_j}}$ ($j = 1, \ldots, n$) as the key/value vector ($k/v$ for short), and a context vector $h_c$ as the query vector ($q$ for short), which is formed by concatenating $\overline{h_\Phi}$ with the first and last nodes of the current solution, i.e., $h_c = [\overline{h_{\phi_1}}, \overline{h_{\phi_{\pi_t}}}]$. The multi-head cross attention operation is then performed and mask nodes that have already been visited.

After calculating the compatibilities $u_{(c)j}$ ($c$ symbolizes the current context node), the decoder proceeds to compute the probabilities for the remaining unmasked nodes. This probability computation is accomplished using a single-head attention mechanism. To scale the compatibility results within the range of $[-\zeta, \zeta]$ ($\zeta = 10$), the *tanh* function is applied. Subsequently, a *softmax* function is utilized to calculate the final output probability vector.

$$u_{(c)j} = \begin{cases} \zeta * \tanh(\frac{q_{(c)} k_j^T}{\sqrt{d_k}}), & \text{if } j \text{ not masked;} \\ -\infty, & \text{otherwise.} \end{cases} \tag{8}$$

$$p_j = p_\theta(\pi_t = j | s, \pi_{1:t-1}) = \frac{e^{u_{(c)j}}}{\sum_{j'=1}^n e^{u_{(c)j'}}}. \tag{9}$$

At last, the decoder selects a node according to the probability vector and proceeds to the next decoding step.

## 4.3 Training

To train the model for RTSP, we adopt a combination of the classic REINFORCE [34] policy gradient algorithm and the POMO training approach [27], which primarily exploits the solution symmetry. We utilize Monte Carlo methods to sample $n$ solution trajectories for each instance, with each trajectory using a different starting node. The reward is computed for each of these $n$ trajectories, and the average reward of the $n$ trajectories is used as a shared baseline within a mini-batch. To maximize the expected return $J$, we employ gradient ascent to approximate $J$, and the Adam optimizer [35] is used to optimize the policy network parameters.

In the following equations, $\pi_i$ represents the generated solution in the $i$-th sampled trajectory, and $R(\pi)$ denotes the reward of the solution as defined in Equation (7).

To efficiently calculate the reward $R(\pi)$, a pretrained "MatNet" is used as a TSP solver. Since a TSP instance can also be represented by a bipartite graph, it is inputted into the MatNet

---

**Algorithm 1** Training process for the RTSP model

**Input**: Sample set $S$, number of epochs $E$, number of steps per epoch $T$, batch size $B$, total number of starting nodes per instance $n$

**Initialization**: policy network parameter $\theta$

1: **for** $epoch \leftarrow 1$ to $E$ **do**
2:    **for** $step \leftarrow 1$ to $T$ **do**
3:       $s_i \leftarrow$ SampleInstance$(S)$    $\forall i \in \{1, ..., B\}$.
4:       $\{st_i^1, st_i^2, ..., st_i^n\} \leftarrow$ StartNodes$(s_i)$
        $\forall i \in \{1, ..., B\}$.
5:       $\pi_i^j \leftarrow$ SampleRollout$(st_i^j, s_i, p_\theta)$
        $\forall i \in \{1, ..., B\}, \forall j \in \{1, ..., n\}$.
6:       $wc_i^j \leftarrow$ WorstCaseTSP$(\pi_i^j)$
        $\forall i \in \{1, ..., B\}, \forall j \in \{1, ..., n\}$.
7:       $R(\pi_i^j) \leftarrow \sum wc_i^j \pi_i^j$ - PretrainedTSP$(wc_i^j)$
        $\forall i \in \{1, ..., B\}, \forall j \in \{1, ..., n\}$.
8:       $b_i \leftarrow \frac{1}{n} \sum_{j=1}^n R(\pi_i^j)$    $\forall i \in \{1, ..., B\}$.
9:       $\nabla_\theta J(\theta) \leftarrow$
        $\frac{1}{Bn} \sum_{i=1}^B \sum_{j=1}^n (R(\pi_i^j) - b_i) \nabla_\theta \log p_\theta(\pi_i^j)$.
10:     $\theta \leftarrow \theta + \alpha \nabla_\theta J(\theta)$.
11:    **end for**
12: **end for**

---

to construct a TSP route. Multiple greedy trajectories with different starting nodes and instance augmentation are utilized to generate TSP solutions.

$$\nabla_\theta J(\theta) \approx \frac{1}{n} \sum_{i=1}^n (R(\pi_i) - b(s)) \nabla_\theta \log_{p_\theta}(\pi_i|s), \qquad b(s) = \frac{1}{n} \sum_{i=1}^n R(\pi_i). \qquad (10)$$

Finally, Algorithm 1 outlines the training process for RTSP. In line 7, we feed the worst-case scenario corresponding to the current solution into the pre-trained TSP model, which provides us with a near-optimal TSP solution as feedback.

### 4.4 Inference

Once the RTSP model is trained, it can be used to make fast inference on a batch of instances. Algorithm 2 presents the overall inference process.

We use the POMO inference approach to sample multi-start solutions. Instance augmentation [30] during inference can also be adjusted to improve the solution quality. Concretely, we employ multiple independent randomizations on the initial one-hot node embeddings $h_{\psi_j}^0$ for each instance to explore various paths towards the optimal solution. Since two matrices are inputted to our model, they share the same randomized one-hot vector. Ultimately, the best solution is selected from the multiple generated solutions.

Recall that our proposed method is still an approximate approach. To ensure a fair and accurate comparison of the final results during the experimental sessions, we adopt a consistent approach by utilizing an exact TSP solver, namely *Gurobi*[2], to determine the target value achieved through different methods (line 8).

---

**Algorithm 2** Inference with the pre-trained TSP model

**Input**: instance $s$, augmentation factor $K$, policy $p_\theta$, total number of starting nodes per instance $n$

**Output**: target solution $\pi^*$, target objective value $obj^*$

1: $\{s_1, ..., s_K\} \leftarrow$ Augment_Onehot_Embeddings$(s)$.
2: $\{st_k^1, ..., st_k^n\} \leftarrow$ StartNodes$(s_k)$
   $\forall k \in \{1, ..., K\}$.
3: $\pi_k^j \leftarrow$ GreedyRollout$(st_k^j, s, p_\theta)$
   $\forall j \in \{1, ..., n\}, \forall k \in \{1, ..., K\}$.
4: $wc_k^j \leftarrow$ WorstCaseTSP$(\pi_k^j)$
   $\forall j \in \{1, ..., n\}, \forall k \in \{1, ..., K\}$.
5: $R(\pi_k^j) \leftarrow \sum wc_k^j \pi_k^j$ - PretrainedTSP$(wc_k^j)$
   $\forall j \in \{1, ..., n\}, \forall k \in \{1, ..., K\}$.
6: $k_{max}, j_{max} \leftarrow argmax_{k,j} R(\pi_k^j)$.
7: $\pi^* \leftarrow \pi_{k_{max}}^{j_{max}}$.
8: $obj^* \leftarrow \sum wc(\pi^*)\pi^* - Gurobi(wc(\pi^*))$.
9: **return** $\pi^*, obj^*$.

# 5   Experiments

Our proposed approach was programmed with Pytorch. All the experiments were conducted on a workstation with Intel(R) Core(TM) i5-4590 CPU @ 3.30GHz, 8.0 GB RAM and TITAN Xp GPU.

## 5.1   Experimental Setup of RTSP

**Experimental Data.**   The experimental data in this study was generated following the work of Montemanni et al. [8]. The data generation process follows a specific rule: a "R-$N$-$M$" type problem is considered, where $N$ represents the number of nodes, and $M$ is a range threshold (for the upper bounds). The control parameter is set to $\Gamma \geq \binom{n}{2}$. In this case, the upper bound travel time for each pair of nodes, denoted as $t_{ij}^+$, is randomly selected from set $\{0, 1, 2, ..., M\}$. The lower bound travel time, denoted as $t_{ij}^-$, is randomly chosen from set $\{0, 1, ..., t_{ij}^+\}$. To maintain consistency, the distance values of the upper and lower bounds are then normalized using a scale factor of $M$. In addition, we ensure the upper and lower bound values to satisfy the triangle inequality, respectively.

**Hyper-parameters.**   During the training process, we follow specific configurations based on the scale of the problem. For small-scale instances with 20 and 30 nodes, training is conducted on a single GPU. The batch size is set to 200 and 100, respectively. We use the Adam optimizer with a learning rate $\alpha = 4 \times 10^{-4}$. Each epoch consists of training on 1000 instances. For larger-scale instances with 40 and 50 nodes, training is performed on two and three GPUs, respectively. The batch size is adjusted to 64 and 32, respectively, to accommodate the GPU memory limitations. The learning rate for the Adam optimizer is set to $\alpha = 2 \times 10^{-4}$. Each GPU handles 400 training instances per epoch. For the built-in TSP models, they are trained for 2000 epochs to ensure their effectiveness and convergence.

**Baselines.**   We have collected typical algorithms for solving RTSP as our baselines. These algorithms can be further categorized into exact and heuristic approaches.

- **Exact approaches:** Branch-and-Cut (BC) and Benders Decomposition (BD) are exact algorithms proposed by Montemanni et al. [8] for solving RTSP. We have implemented these algorithms using the Python programming language and the Gurobi optimizer. For small-scale problems, such as those with 20 nodes, these two algorithms exhibit fast solution speeds. However, as the problem size increases, for example, with 50 nodes, the computation time required to obtain a solution is significantly increased.

- **Heuristic approaches:** There are three heuristic baselines introduced for comparisons: heuristic algorithms based on simulated annealing (SA-based) [10], iterated dual substitution method (iDS) [36], and edge generation algorithm (EGA) [11]. To ensure fairness, we utilize the source codes provided by the authors, with the exception of SA-based, which we re-implemented ourselves. Furthermore, we maintain most of the original settings of these algorithms.

The time limit of all above algorithms are set to 3600 seconds for every instance, except that the SA-based method has a time limit of $15n$ due to the limitation of repeated calls of the LKH algorithm. Nevertheless, some algorithms may stop earlier when reaching their termination criteria.

For the experimental setting of RCVRP, please refer to Appendix 7.5.

## 5.2   Results and Discussions

In the following reported tables, "Obj" represents the average objective value accurately computed using *Gurobi* to solve the worst-case routing scenario. "Gap" indicates the relative difference between the objective value obtained by each algorithm and the best-known optimal value, expressed as a percentage: $Gap = \frac{Obj - Obj_{best}}{Obj_{best}} \times 100\%$. "Time" represents the average solving time per instance.

**Comparison Results of RTSP.**   Table 1 presents the main results of our trained model and other methods on randomly generated instances. The methods are categorized into three groups: exact methods, heuristic methods, and our proposed methods. The results of our methods are reported

Table 1: Computational results on generated RTSP instances.

| Method | Obj | Gap | Time(s) | Obj | Gap | Time(s) | Obj | Gap | Time(s) | Obj | Gap | Time(s) |
|---|---|---|---|---|---|---|---|---|---|---|---|---|
| | R-20-10 | | | R-30-10 | | | R-40-10 | | | R-50-10 | | |
| BC | 4.79 | 0.00% | 2.3 | 7.23 | 0.00% | 10.3 | 9.14 | 0.00% | 12.1 | 11.36 | 0.00% | 135.9 |
| BD | 4.79 | 0.00% | 26.9 | 7.23 | 0.00% | 71.3 | 9.14 | 0.00% | 107.7 | 11.36 | 0.00% | 626.5 |
| SA-based | 4.88 | 1.88% | 300.1 | 7.45 | 3.04% | 450.2 | 9.69 | 6.02% | 600.4 | 12.11 | 6.60% | 750.5 |
| iDS | 4,79 | 0.00% | 3600.0 | 7.23 | 0.00% | 3600.0 | 9.14 | 0.00% | 3600.1 | 11.37 | 0.09% | 3600.1 |
| EGA | 4.79 | 0.00% | 1682.7 | 7.23 | 0.00% | 3037.9 | 9.14 | 0.00% | 3452.3 | 11.36 | 0.00% | 3617.0 |
| ours. | 4.80 | 0.21% | **0.3** | 7.35 | 1.66% | **0.4** | 9.30 | 1.75% | **0.5** | 11.65 | 2.55% | **0.9** |
| ours,$\times 8$. | 4.79 | 0.00% | 0.7 | 7.32 | 1.24% | 1.5 | 9.21 | 0.77% | 2.8 | 11.52 | 1.41% | 5.5 |
| ours,$\times 128.(n=50,\times 64)$ | 4.79 | 0.00% | 9.5 | 7.26 | 0.41% | 23.2 | 9.16 | 0.22% | 49.3 | 11.44 | 0.70% | 45.8 |
| | R-20-100 | | | R-30-100 | | | R-40-100 | | | R-50-100 | | |
| BC | 4.078 | 0.00% | 11.1 | 6.008 | 0.00% | 553.8 | 7.824 | 0.06% | 2978.2 | 9.717 | 0.64% | 1466.7 |
| BD | 4.078 | 0.00% | 392.9 | 6.047 | 0.65% | 3448.9 | 7.883 | 0.82% | 3600.1 | 9.693 | 0.39% | 3600.2 |
| SA-based | 4.142 | 1.57% | 300.0 | 6.234 | 3.76% | 450.1 | 8.288 | 5.14% | 600.2 | 10.546 | 9.23% | 750.3 |
| iDS | 4.078 | 0.00% | 3600.0 | 6.020 | 0.20% | 3600.1 | 7.849 | 0.38% | 3600.3 | 9.670 | 0.16% | 3601.4 |
| EGA | 4.078 | 0.00% | 1407.0 | 6.008 | 0.00% | 3605.9 | 7.819 | 0.00% | 3623.3 | 9.655 | 0.00% | 3636.4 |
| ours. | 4.100 | 0.54% | **0.3** | 6.116 | 1.80% | **0.4** | 7.915 | 1.23% | **0.6** | 9.847 | 1.99% | **0.8** |
| ours,$\times 8$. | 4.079 | 0.02% | 0.7 | 6.039 | 0.52% | 1.5 | 7.871 | 0.67% | 2.9 | 9.767 | 1.16% | 5.3 |
| ours,$\times 128.(n=50,\times 64)$ | 4.079 | 0.02% | 9.6 | 6.017 | 0.15% | 23.7 | 7.828 | 0.12% | 50.1 | 9.707 | 0.54% | 45.6 |
| | R-20-1000 | | | R-30-1000 | | | R-40-1000 | | | R-50-1000 | | |
| BC | 3.8366 | 0.00% | 7.9 | 6.1996 | 0.00% | 835.8 | 7.8578 | 0.40% | 2941.8 | 9.9995 | 1.79% | 3600.3 |
| BD | 3.8366 | 0.00% | 270.8 | 6.2582 | 0.95% | 3329.9 | 7.8859 | 0.76% | 3600.1 | 9.9017 | 0.80% | 3600.1 |
| SA-based | 3.8679 | 0.82% | 300.0 | 6.4963 | 4.79% | 450.1 | 8.3930 | 6.43% | 600.2 | 10.6223 | 8.13% | 752.2 |
| iDS | 3.8386 | 0.05% | 3600.0 | 6.2153 | 0.25% | 3600.1 | 7.8568 | 0.38% | 3600.3 | 9.8737 | 0.51% | 3669.6 |
| EGA | 3.8366 | 0.00% | 1406.6 | 6.1996 | 0.00% | 3604.8 | 7.8268 | 0.00% | 3618.1 | 9.8234 | 0.00% | 3658.2 |
| ours. | 3.8608 | 0.63% | **0.3** | 6.2740 | 1.20% | **0.4** | 7.9697 | 1.80% | **0.6** | 9.9870 | 1.67% | **0.8** |
| ours,$\times 8$. | 3.8366 | 0.00% | 0.7 | 6.2229 | 0.38% | 1.5 | 7.9084 | 1.04% | 2.9 | 9.8948 | 0.73% | 5.4 |
| ours,$\times 128.(n=50,\times 64)$ | 3.8366 | 0.00% | 9.7 | 6.2021 | 0.04% | 23.9 | 7.8516 | 0.32% | 49.7 | 9.8536 | 0.31% | 45.3 |

for three versions: no augmentation, augmentation with 8 instances, and augmentation with 128 instances (except 64 instances for 50 nodes due to memory limitations).

From Table 1, it is evident that our approach consistently achieves high-quality solutions in significantly small time. Without instance augmentation, the gaps obtained by our method are consistently below 2.6%. When utilizing instance augmentation, our method can obtain approximately accurate or even exact solutions for small scales (such as 20 nodes). For larger scales, our method also produces solutions with gaps less than 0.7%. In terms of runtime, under both scenarios with and without instance augmentation, our approach has a large and significant advantage over exact methods and heuristics, taking only a fraction of a second. The results verify the effectiveness of our approach.

**Generalization Results of RTSP.** We conduct experiments to evaluate the generalization ability of our model, specifically its performance on test instances that are outside the training distribution. To illustrate this, we consider an example where $M = 100$ and the augmentation involves 8 instances. We evaluate our models, trained with varying numbers of nodes, on test data with different node sizes. The results are depicted in Figure 3. As expected, the model trained on the corresponding node size achieves the best performance on the test data. Moreover, the model trained on 50 nodes demonstrates competitive results when tested on the other three scales ($N = 20, 30, 40$), suggesting a

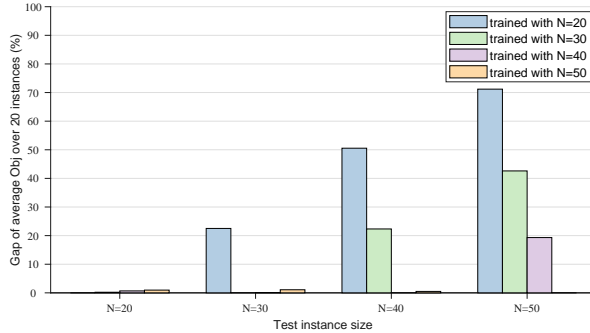

Figure 3: Performance on varying-scale problems. Gap is relative to the test results of the models trained on the consistent scales.

Table 2: Results of different encoding methods for the uncertainty sets' upper and lower bounds on R-40-100.

| Method | No augmentation | | Augmentation (×8) | | Augmentation (×128) | |
|---|---|---|---|---|---|---|
| | Obj | Gap | Obj | Gap | Obj | Gap |
| ours | 7.9610 | **1.55%** | 7.8985 | **0.75%** | 7.8600 | **0.26%** |
| blended | 8.0640 | 2.86% | 7.9095 | 0.89% | 7.8760 | 0.47% |
| fusion | 8.0080 | 2.15% | 7.9045 | 0.83% | 7.8645 | 0.32% |

Table 3: Computational results for RCVRP.

| Method | Obj | Gap | Time($s$) | Obj | Gap | Time($s$) | Obj | Gap | Time($s$) | Obj | Gap | Time($s$) |
|---|---|---|---|---|---|---|---|---|---|---|---|---|
| | R-20-100 | | | R-20-1000 | | | R-50-100 | | | R-50-1000 | | |
| BC | 5.2855 | 0.46% | 3605.5 | 4.87385 | 1.75% | 3607.8 | 20.6335 | 73.35% | 3804.4 | 20.30060 | 77.13% | 3856.6 |
| BD | 5.2130 | 0.99% | 3663.1 | 4.86765 | 1.62% | 3662.9 | 12.3385 | 3.66% | 3961.1 | 12.17785 | 6.26% | 3960.9 |
| ours. | 5.3130 | 2.93% | **0.3** | 4.96790 | 3.72% | **0.3** | 12.2270 | 2.72% | **0.9** | 11.95765 | 4.33% | **1.0** |
| ours,×8. | 5.1810 | 0.37% | 0.8 | 4.81545 | 0.53% | 0.8 | 11.9395 | 0.31% | 6.3 | 11.62230 | 1.41% | 6.2 |
| ours,×32. | 5.1620 | 0.00% | 2.7 | 4.78990 | 0.00% | 2.8 | 11.9030 | 0.00% | 25.6 | 11.46090 | 0.00% | 25.9 |

strong generalization capability. Conversely, the models trained on smaller scales exhibit weaker generalization capability.

**Ablation Studies of RTSP.** We investigate the impact of different designs for feature encoding of the uncertainty support set. We explore three encoding methods, namely "ours", "blended"(a blended matrix is computed by the weighted sum of the upper and lower bound matrices, which is then encoded by MatNet) and "fusion"(two distance matrices and attention scores are fused through an MLP in the multi-head mixed score attention layer of the encoder), with their architectures depicted in Figure 5 in Appendix 7.4.

We compare these encoding methods on R-40-100 with 20 random instances. As shown in Table 2, our method outperforms both "blended" and "fusion" methods in all cases, including scenarios without instance augmentation, as well as those with ×8 and ×128 instance augmentation.

**Comparison Results of RCVRP.** Table 3 shows the comparison results between our method and the exact algorithm. The results are all obtained using the Gurobi solver in a uniform manner. To our knowledge, this specific type of RCVRP has never been discussed before. The results show that our algorithm still achieves high solution quality and has a clear advantage in the solution time of RCVRP. On a 20-node scale, our gap is around 0.5%. On a 50-node scale, our gap is below 4.5%. In terms of time consumption, we only need a few tenths of a second to dozens of seconds.

# 6 Conclusions

This paper focuses on addressing the robust routing problem with uncertain travel times, relevant to real-world scenarios. We introduce a neural combinatorial optimization approach, employing an end-to-end model with attention mechanism for automatic policy learning. Leveraging a pre-trained model, experimental results demonstrate that our method can generate near-optimal solutions much faster than exact and heuristic approaches. However, due to hardware memory limitations and constraints of the POMO algorithm used in training, our method is currently limited in scale and to problems satisfying the max-regret theorem. Moving forward, we aim to expand our approach to handle larger-scale problems and tackle a broader range of robust optimization problems.

# Acknowledgements and Disclosure of Funding

This work is supported by the National Natural Science Foundation of China (62072483) and Guangdong Natural Science Funds (2024A1515010871).

## Footnotes

[1] https://github.com/xchihiro/Robust-VRP

[2]https://pypi.org/project/gurobipy

# References

[1] Panos Kouvelis and Gang Yu. *Robust discrete optimization and its applications*, volume 14. Springer Science & Business Media, 2013.

[2] Richard E Bellman and Lotfi Asker Zadeh. Decision-making in a fuzzy environment. *Management science*, 17(4):B–141, 1970.

[3] Aharon Ben-Tal, Laurent El Ghaoui, and Arkadi Nemirovski. *Robust optimization*, volume 28. Princeton university press, 2009.

[4] Stephen P Boyd and Lieven Vandenberghe. *Convex optimization*. Cambridge university press, 2004.

[5] Dimitris Bertsimas and Aurélie Thiele. Robust and data-driven optimization: modern decision making under uncertainty. In *Models, methods, and applications for innovative decision making*, pages 95–122. INFORMS, 2006.

[6] Dimitris Bertsimas and Melvyn Sim. The price of robustness. *Operations research*, 52(1): 35–53, 2004.

[7] Aharon Ben-Tal, Alexander Goryashko, Elana Guslitzer, and Arkadi Nemirovski. Adjustable robust solutions of uncertain linear programs. *Mathematical programming*, 99(2):351–376, 2004.

[8] Roberto Montemanni, János Barta, Monaldo Mastrolilli, and Luca Maria Gambardella. The robust traveling salesman problem with interval data. *Transportation Science*, 41(3):366–381, 2007.

[9] André Chassein and Marc Goerigk. On the recoverable robust traveling salesman problem. *Optimization Letters*, 10:1479–1492, 2016.

[10] Chung-Cheng Lu, Shih-Wei Lin, and Kuo-Ching Ying. Minimizing worst-case regret of makespan on a single machine with uncertain processing and setup times. *Applied Soft Computing*, 23:144–151, 2014.

[11] K Hasegawa and W Wu. A heuristic approach for the robust traveling salesman problem. In *2022 IEEE International Conference on Industrial Engineering and Engineering Management (IEEM)*, pages 0561–0565. IEEE, 2022.

[12] Ilgaz Sungur, Fernando Ordónez, and Maged Dessouky. A robust optimization approach for the capacitated vehicle routing problem with demand uncertainty. *Iie Transactions*, 40(5):509–523, 2008.

[13] Elyn Solano-Charris, Christian Prins, and Andréa Cynthia Santos. Local search based meta-heuristics for the robust vehicle routing problem with discrete scenarios. *Applied Soft Computing*, 32:518–531, 2015.

[14] Lars Eufinger, Jannis Kurtz, Christoph Buchheim, and Uwe Clausen. A robust approach to the capacitated vehicle routing problem with uncertain costs. *INFORMS Journal on Optimization*, 2(2):79–95, 2020.

[15] Aigerim Bogyrbayeva, Meraryslan Meraliyev, Taukekhan Mustakhov, and Bissenbay Dauletbayev. Machine learning to solve vehicle routing problems: A survey. *IEEE Transactions on Intelligent Transportation Systems*, 2024.

[16] Marijn Van Knippenberg, Mike Holenderski, and Vlado Menkovski. Complex vehicle routing with memory augmented neural networks. In *2020 IEEE Conference on Industrial Cyberphysical Systems (ICPS)*, volume 1, pages 303–308. IEEE, 2020.

[17] Chaitanya K Joshi, Thomas Laurent, and Xavier Bresson. An efficient graph convolutional network technique for the travelling salesman problem. *arXiv preprint arXiv:1906.01227*, 2019.

[18] Chaitanya K Joshi, Quentin Cappart, Louis-Martin Rousseau, and Thomas Laurent. Learning the travelling salesperson problem requires rethinking generalization. *Constraints*, 27(1):70–98, 2022.

[19] Elias Khalil, Hanjun Dai, Yuyu Zhang, Bistra Dilkina, and Le Song. Learning combinatorial optimization algorithms over graphs. *Advances in neural information processing systems*, 30, 2017.

[20] Volodymyr Mnih, Koray Kavukcuoglu, David Silver, Andrei A Rusu, Joel Veness, Marc G Bellemare, Alex Graves, Martin Riedmiller, Andreas K Fidjeland, Georg Ostrovski, et al. Human-level control through deep reinforcement learning. *nature*, 518(7540):529–533, 2015.

[21] Martin Riedmiller. Neural fitted q iteration–first experiences with a data efficient neural reinforcement learning method. In *Machine learning: ECML 2005: 16th European conference on machine learning, Porto, Portugal, October 3-7, 2005. proceedings 16*, pages 317–328. Springer, 2005.

[22] Oriol Vinyals, Meire Fortunato, and Navdeep Jaitly. Pointer networks. *Advances in Neural Information Processing Systems*, 28, 2015.

[23] Irwan Bello, Hieu Pham, Quoc V Le, Mohammad Norouzi, and Samy Bengio. Neural combinatorial optimization with reinforcement learning. *arXiv preprint arXiv:1611.09940*, 2016.

[24] Mohammadreza Nazari, Afshin Oroojlooy, Lawrence Snyder, and Martin Takác. Reinforcement learning for solving the vehicle routing problem. *Advances in neural information processing systems*, 31, 2018.

[25] Wouter Kool, Herke Van Hoof, and Max Welling. Attention, learn to solve routing problems! *arXiv preprint arXiv:1803.08475*, 2018.

[26] Ashish Vaswani, Noam Shazeer, Niki Parmar, Jakob Uszkoreit, Llion Jones, Aidan N Gomez, Łukasz Kaiser, and Illia Polosukhin. Attention is all you need. *Advances in neural information processing systems*, 30, 2017.

[27] Yeong-Dae Kwon, Jinho Choo, Byoungjip Kim, Iljoo Yoon, Youngjune Gwon, and Seungjai Min. Pomo: Policy optimization with multiple optima for reinforcement learning. *Advances in Neural Information Processing Systems*, 33:21188–21198, 2020.

[28] Zizhen Zhang, Hong Liu, MengChu Zhou, and Jiahai Wang. Solving dynamic traveling salesman problems with deep reinforcement learning. *IEEE Transactions on Neural Networks and Learning Systems*, 34(4):2119–2132, 2021.

[29] Yongxin Zhang, Jiahai Wang, and Zizhen Zhang. Edge-based formulation with graph attention network for practical vehicle routing problem with time windows. In *2022 International Joint Conference on Neural Networks (IJCNN)*, pages 01–08. IEEE, 2022.

[30] Yeong-Dae Kwon, Jinho Choo, Iljoo Yoon, Minah Park, Duwon Park, and Youngjune Gwon. Matrix encoding networks for neural combinatorial optimization. *Advances in Neural Information Processing Systems*, 34:5138–5149, 2021.

[31] Jianan Zhou, Yaoxin Wu, Wen Song, Zhiguang Cao, and Jie Zhang. Towards omni-generalizable neural methods for vehicle routing problems. *arXiv preprint arXiv:2305.19587*, 2023.

[32] Tobias Jacobs, Francesco Alesiani, and Gülcin Ermis. Reinforcement learning for route optimization with robustness guarantees. In *IJCAI*, pages 2592–2598, 2021.

[33] Justin Dumouchelle, Esther Julien, Jannis Kurtz, and Elias B Khalil. Neur2ro: Neural two-stage robust optimization. *arXiv preprint arXiv:2310.04345*, 2023.

[34] Ronald J Williams. Simple statistical gradient-following algorithms for connectionist reinforcement learning. *Machine learning*, 8:229–256, 1992.

[35] Diederik P Kingma and Jimmy Ba. Adam: A method for stochastic optimization. *arXiv preprint arXiv:1412.6980*, 2014.

[36] Wei Wu, Manuel Iori, Silvano Martello, and Mutsunori Yagiura. An iterated dual substitution approach for binary integer programming problems under the min-max regret criterion. *INFORMS Journal on Computing*, 34(5):2523–2539, 2022.

[37] Clair E Miller, Albert W Tucker, and Richard A Zemlin. Integer programming formulation of traveling salesman problems. *Journal of the ACM (JACM)*, 7(4):326–329, 1960.

[38] Keld Helsgaun. An effective implementation of the lin–kernighan traveling salesman heuristic. *European journal of operational research*, 126(1):106–130, 2000.

[39] Nikolaus Hansen. The cma evolution strategy: A tutorial. *arXiv preprint arXiv:1604.00772*, 2016.

# 7 Appendix

## 7.1 Proof of Theorem 1

*Proof.* To provide a coherent description of the proof process, we first introduce a shorthand notation for certain equations.

1. The optimal tour under scenario $u$ is denoted as $y^*(u)$.

2. The worst case scenario obtained by solution $x$ according to the theorem is denoted as $wc(x)$.

3. The tour length of solution $x$ under scenario $u$ is denoted as $C(u,x)$. By the definition of regret, the value of regret for solution $x$ under scenario $u$ has the equation: $regret(u,x) = C(u,x) - C(u,y^*(u))$.

We then outline a concise proof procedure. Assume that we have a solution $x$, our goal is to prove that the regret value under scenario $wc(x)$ is greater than or equal to the regret value under any other scenario, i.e., $regret(u,x) \leq regret(wc(x),x)$. Here,

$$
\begin{aligned}
regret(u,x) &= C(u,x) - C(u,y^*(u)) \\
&= \sum_{(i,j)\in x} t_{ij}^u - \sum_{(i,j)\in y^*(u)} t_{ij}^u \\
&= \sum_{(i,j)\in x\backslash y^*(u)} t_{ij}^u - \sum_{(i,j)\in y^*(u)\backslash x} t_{ij}^u.
\end{aligned}
\tag{11}
$$

Based on the definition of $wc(x)$, it is evident that when $(i,j) \in x \setminus y^*(u), t_{ij}^{wc(x)} \geq t_{ij}^u$. Similarly, when $(i,j) \in y^*(u) \setminus x, t_{ij}^{wc(x)} \leq t_{ij}^u$. We have:

$$
\begin{aligned}
regret(u,x) &= C(u,x) - C(u,y^*(u)) \\
&\leq \sum_{(i,j)\in x\backslash y^*(u)} t_{ij}^{wc(x)} - \sum_{(i,j)\in y^*(u)\backslash x} t_{ij}^{wc(x)} \\
&= \sum_{(i,j)\in x} t_{ij}^{wc(x)} - \sum_{(i,j)\in x\cap y^*(u)} t_{ij}^{wc(x)} - \sum_{(i,j)\in y^*(u)\backslash x} t_{ij}^{wc(x)} \\
&= \sum_{(i,j)\in x} t_{ij}^{wc(x)} - (\sum_{(i,j)\in x\cap y^*(u)} t_{ij}^{wc(x)} + \sum_{(i,j)\in y^*(u)\backslash x} t_{ij}^{wc(x)}) \\
&= \sum_{(i,j)\in x} t_{ij}^{wc(x)} - (\sum_{(i,j)\in y^*(u)} t_{ij}^{wc(x)}) \\
&= C(wc(x),x) - C(wc(x),y^*(u)) \\
&\leq C(wc(x),x) - C(wc(x),y^*(wc(x))) \\
&= regret(wc(x),x),
\end{aligned}
\tag{12}
$$

Therefore, the proof of Theorem 1 is concluded.

$\square$

## 7.2 RTSP Example with budget uncertainty

In Figure 4, we provide a specific example to illustrate the solution process of RTSP with budget uncertainty while $\Gamma = \left\lfloor \frac{\binom{n}{2}}{4} \right\rfloor$. Recall that the budget uncertainty set is defined as $\mathbb{U}_{budget} = \{t|t_{ij} = t_{ij}^- + \hat{t}_{ij}\eta_{ij}, 0 \leq \eta_{ij} \leq 1, \forall (i,j) \in E; \sum_{(i,j)\in E} \eta_{ij} \leq \Gamma\}$. where $\hat{t}_{ij} = t_{ij}^+ - t_{ij}^-$. Figure 4 depicts a graph with 4 nodes and 6 edges. Suppose the values of $t_{ij}^-$ of each edge are as shown in Figure 4a. For simplicity, let's assume that all $\hat{t}_{ij}$ values are set to the same value 2, though in a real

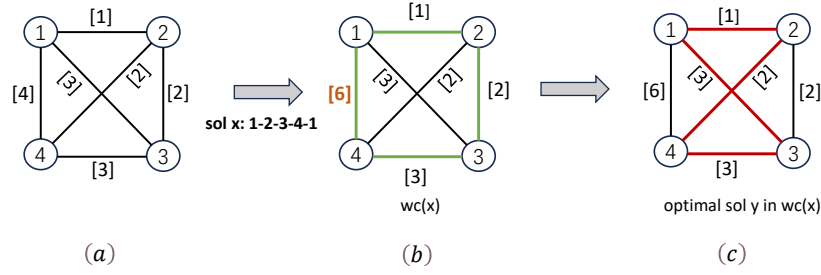

Figure 4: An instance of RTSP with budget uncertainty and $\Gamma = \left\lfloor \frac{\binom{n}{2}}{4} \right\rfloor$. (a) $t_{ij}^-$. (b) Worst case scenario $wc(x)$ corresponding to route $x$. (c) Optimal TSP solution under scenario $wc(x)$.

implementation, these values would differ. Consider a feasible solution "$1 \rightarrow 2 \rightarrow 3 \rightarrow 4 \rightarrow 1$" highlighted by the green edges in Figure 4b, under the given $\Gamma$ ,the worst scenario corresponding to this solution is also shown in Figure 4b, with a nominal total time is $1 + 2 + 3 + 6 = 12$. The optimal solution in this scenario, highlighted in red in Figure 4c, is either the route $1 \rightarrow 2 \rightarrow 4 \rightarrow 3 \rightarrow 1$ or $1 \rightarrow 3 \rightarrow 4 \rightarrow 2 \rightarrow 1$, both of which have a total cost of 10. The max-regret value for the original solution is calculated as $12 - 10 = 2$. By evaluating the max-regret for each feasible solution, the final optimal solution is identified as the one with the smallest max-regret value.

## 7.3 Formulations of Robust Routing Problems

**RTSP Formulation**   Based on the TSP formulation and Corollary 1, we present the following mathematical model for RTSP with the min-max-regret criterion. This modeling serves as the foundation for both exact algorithms and heuristic algorithms in this field.

The mathematical scenario for RTSP is defined as follows. Given a complete symmetric graph $G = (V, E)$, where $V = \{1, 2, ..., n\}$ is the set of nodes. The travel time (cost) of an edge $(i, j)$ is defined by an interval uncertainty set $[t_{ij}^-, t_{ij}^+]$, where $t_{ij}^-$ and $t_{ij}^+$ denote the lower and upper bounds of possible values, respectively.

$$\min \quad \sum_{(i,j) \in E} t_{ij}^+ x_{ij} - r \tag{13}$$

$$\text{s.t.} \quad r \leq \sum_{(i,j) \in E} y_{ij} t_{ij}^- + \sum_{(i,j) \in E} y_{ij}(t_{ij}^+ - t_{ij}^-) x_{ij}, \quad \forall y \in \mathbb{S} \tag{14}$$

$$x \in \mathbb{S} \tag{15}$$

$$r \in \mathcal{R} \tag{16}$$

In the above model, $x_{ij}$ are the decision variables for the RTSP solution, while $y_{ij}$ serves as temporary auxiliary decision variables representing the decision variable for TSP in the worst-case scenario corresponding to each $x$. To establish the nonlinear interaction between $x_{ij}$ and $y_{ij}$, a free variable $r$ is introduced.

The objective function in Equation (13) aims to minimize the maximum regret value, while Constraint (14) ensures the maximization of regret. We define the set of all feasible TSP solutions as $\mathbb{S}$, which is

bounded by the following constraints.

$$\sum_{i \in V, i \neq j} x_{ij} = 1, \qquad \forall j \in V \tag{17}$$

$$\sum_{j \in V, j \neq i} x_{ij} = 1, \qquad \forall i \in V \tag{18}$$

$$\sum_{i,j \in S} x_{ij} \leq |S| - 1, \quad \forall S \subsetneq V, S \neq \emptyset \tag{19}$$

$$x_{ij} \in \{0, 1\}, \qquad \forall (i, j) \in E \tag{20}$$

Constraint (15) guarantee the feasibility of $x_{ij}$ as a TSP solution. It is important to note that $y_{ij}$ must also be a feasible and valid TSP solution, and thus we have $y \in \mathbb{S}$ in Equation (14).

**RCVRP Formulation** CVRP is an extension of TSP that additionally considers vehicle capacity constraints. Similar to RTSP, we can use the mathematical model of the nominal CVRP and Corollary 1 to extend the mathematical model of RCVRP.

The RCVRP scenario is defined as follows. We have a fleet of vehicles with a capacity of $Q$ per vehicle, and $n$ customers distributed geographically. Each customer has its own demand $d_i$ (where $i = 1, 2, ..., n$). Each customer can only be served by one vehicle, and there is no limit on the number of available vehicles. Let $V$ be the set of $n$ demand nodes and a single depot, represented as node 0. The set $E$ represents the edge connections between nodes. Similar to RTSP, we consider the robust scenario where the travel time is subject to interval uncertainty. The travel time (cost) of an edge $(i, j)$ is within the interval $[t_{ij}^-, t_{ij}^+]$, where $t_{ij}^-$ and $t_{ij}^+$ represent the lower and upper bounds of possible values, respectively. Note that in our setting, we assume that $t_{ij} = t_{ji}$, meaning that the travel time matrix is symmetric.

$$\min \quad \sum_{(i,j) \in E} t_{ij}^+ x_{ij} - r \tag{21}$$

$$\text{s.t.} \quad r \leq \sum_{(i,j) \in E} y_{ij} t_{ij}^- + \sum_{(i,j) \in E} y_{ij}(t_{ij}^+ - t_{ij}^-) x_{ij}, \quad \forall y \in \mathbb{S} \tag{22}$$

$$x \in \mathbb{S} \tag{23}$$

$$r \in \mathcal{R} \tag{24}$$

The model is the same as the one described in Section 7.3. It should be noted that $\mathbb{S}$ is defined as the set of all feasible solutions for CVRP, subject to the following constraints.

$$\sum_{i \in V, i \neq j} x_{ij} = 1, \qquad \forall j \in V \setminus \{0\} \tag{25}$$

$$\sum_{j \in V, j \neq i} x_{ij} = 1, \qquad \forall i \in V \setminus \{0\} \tag{26}$$

$$u_j - u_i + Q(1 - x_{ij}) \geq d_j, \quad \forall i, j \in V \setminus \{0\}, i \neq j \tag{27}$$

$$d_i \leq u_i \leq Q, \qquad \forall i \in V \setminus \{0\} \tag{28}$$

$$x_{ij} \in \{0, 1\}, \qquad \forall (i, j) \in E \tag{29}$$

The main difference between CVRP and TSP lies in the consideration of capacity constraints. In CVRP, it is crucial to ensure that the total demand of customers served on each route does not exceed the vehicle's capacity. To address this, the Miller-Tucker-Zemlin (MTZ) constraint [37] introduces continuous variables $u_i$ for every $i \in V \setminus \{0\}$, which represent the flow in the vehicle after it visits customer $i$. Constraints (27)–(28) enforce both the capacity and connectivity requirements of feasible routes.

## 7.4 Additional Results of RTSP

**Generalization on the Threshold Value.** In addition to considering the generalization on dimension $N$, as depicted in Figure 3, we also test the generalization ability on dimension $M$. The models

Table 4: Performance on varying M-threshold problems among instances of $N = 20$, with $\times 8$ instance augmentation.

| Train threshold ＼ Test threshold | $M$=10 | $M$=100 | $M$=1000 |
|---|---|---|---|
| Trained with $M$=10 | 4.8100 | 4.0010 | 4.0762 |
| Trained with $M$=100 | 4.9150 | 3.9895 | 4.0621 |
| Trained with $M$=1000 | 4.8800 | 4.0105 | 4.0637 |

Table 5: Results of various built-in TSP solving algorithms on R-20-100. The training time refers to the average training time per epoch.

| Method | $\times 1$ Obj | $\times 1$ Gap | $\times 8$ Obj | $\times 8$ Gap | $\times 128$ Obj | $\times 128$ Gap | Training time |
|---|---|---|---|---|---|---|---|
| ours | 4.0140 | 0.63% | 3.9895 | **0.01%** | 3.9895 | **0.01%** | $\approx$ 1 min / epoch |
| CMA-ES | 4.0430 | 1.34% | 3.9925 | 0.08% | 3.9895 | **0.01%** | $\approx$ 6 min / epoch |
| LKH | 4.0135 | **0.60%** | 3.9905 | 0.04% | 3.9895 | **0.01%** | $\approx$ 7.5 min / epoch |

trained with different ranges of $M$ are respectively used to test the instances across all ranges of $M$. As shown in the Table 4, it can be seen that a model trained within a specific threshold value can achieve relatively ideal experimental results within other threshold values. This shows that under the premise of fixed $N$, our model can achieve good generalization results for different $M$ values.

**Effects of the Encoding Approaches.**    In Figure 5(a) (labeled as "ours"), our proposed approach is depicted. It involves encoding the upper and lower bound matrices separately using MatNet and then adding the embeddings of the corresponding nodes. In Figure 5(b) (labeled as "blended"), a blended matrix is computed by taking the weighted sum of the upper and lower bound matrices, represented as $D = w * D^{up} + (1.0 - w) * D^{low}$. Here, $w$ is initialized to 0.5 and serves as a learnable parameter. The blended matrix is subsequently encoded using MatNet.Figure 5(c) (labeled as "fusion") illustrates the third approach, which involves fusing the two matrices and attention scores through a multi-layer perceptron (MLP) in the multi-head mixed-score attention layer within the MatNet encoder.

**Effects of the Built-in TSP Solvers.**    We also explore different built-in TSP solvers under the worst-case scenario. In addition to the pre-trained TSP model used in our method, we further consider the well-known heuristic algorithms like LKH [38] and CMA-ES [39]. The optimality gap and training time are reported in Table 5.

Even though LKH is the state-of-the-art TSP solver that can almost find the optimal solution in small-size cases, it has no superiority with regard to solution quality for training an RTSP model and even spends more training time. On the other hand, although CMA-ES evolutionary algorithm could be accelerated through parallelization techniques in a large number of model training processes, it faced a trade-off between solving speed and solution quality. As can be seen, the pre-trained TSP model used in our method exhibits almost the smallest optimality gap and the shortest training time compared with other methods.

**Training Details.**    In Figure 6, we present the training progress curves that illustrate the loss and objective values for "R-50-100" instances, serving as an illustrative example. The figure demonstrates that the training loss steadily converges, and the training score is minimized after 1000 epochs. These observations indicate that the model successfully learns the implicit features of RTSP during the training process.

## 7.5  Experimental Setup of RCVRP

**Data Generation**    We generate random instances of RCVRP for two different sizes: $n = 20$ and $n = 50$. The control parameter is set to $\Gamma \geq \binom{n}{2}$. The following settings are used.

- The vehicle capacity is fixed at $Q = 1.0$.

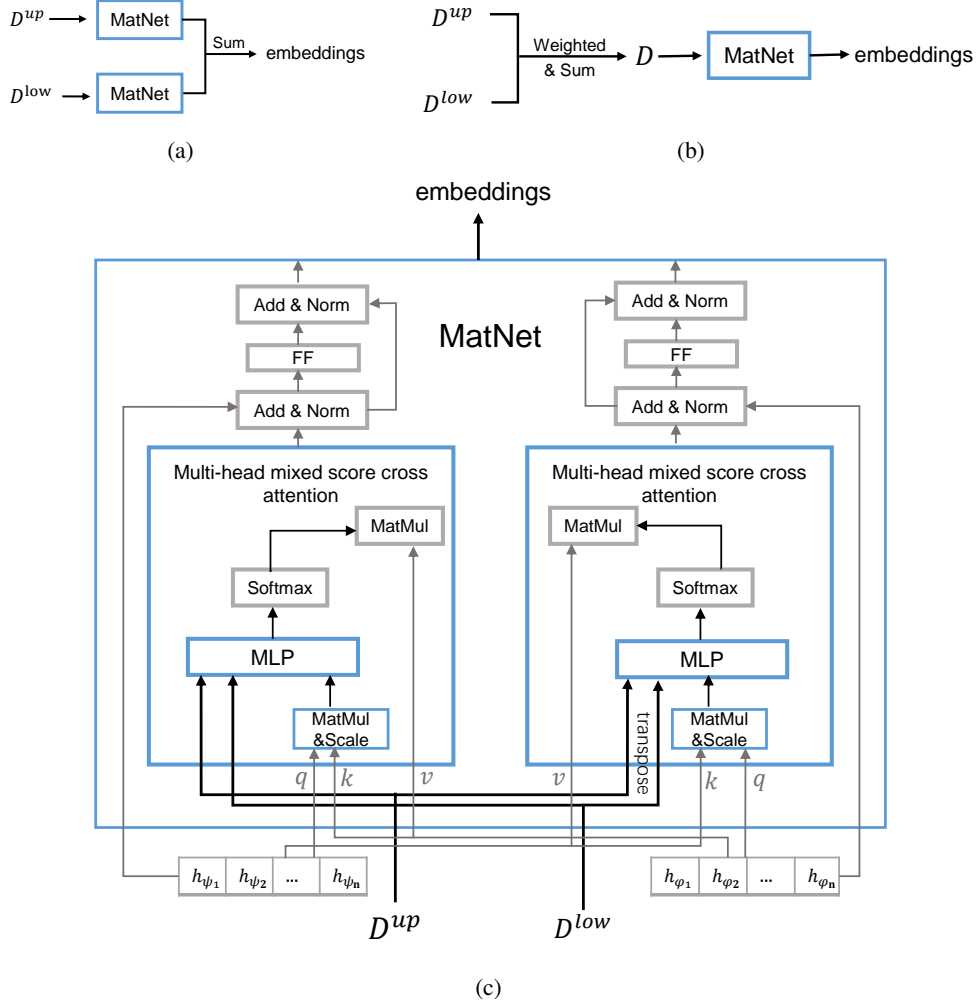

Figure 5: Different encoding methods for the uncertainty set. (a) ours. (b) blended. (c) fusion.

- The demand $d_i$ for each node $i$ is determined by dividing a uniformly sampled value $\hat{d}_i$ from the set $\{1, 2, ..., 9\}$ by a value $D$. For $n = 20$, $D$ is set to 30, while for $n = 50$, $D$ is set to 40.

- The uncertain variable $t_{ij}$ represents the travel time between nodes $i$ and $j$. The upper bound travel time $t_{ij}^+$ is randomly selected from the set $\{0, 1, 2, ..., M\}$, while the lower bound travel time $t_{ij}^-$ is randomly chosen from the set $\{0, 1, ..., t_{ij}^+\}$.

- To ensure consistency, the distance values of the upper and lower bounds are normalized using a scale factor of $M$.

In the data generation process, we also consider the triangle inequality. This inequality is independently applied to both the upper bound and lower bound. For example, for each triplet $(i, j, k)$ in the upper bound, we check if $t_{ik}^+ + t_{kj}^+ > t_{ij}^+$ is violated. If the inequality is violated, we replace $t_{ij}^+$ with $t_{ik}^+ + t_{kj}^+$ to ensure that the triangle inequality holds.

**Hyper-parameters** During the training process, we adopt specific configurations based on the problem scale. For small-scale instances with 20 nodes, training is conducted on a single GPU. The batch size is set to 200. We utilize the Adam optimizer with a learning rate of $\alpha = 4 \times 10^{-4}$. Each epoch involves training on 1000 instances.

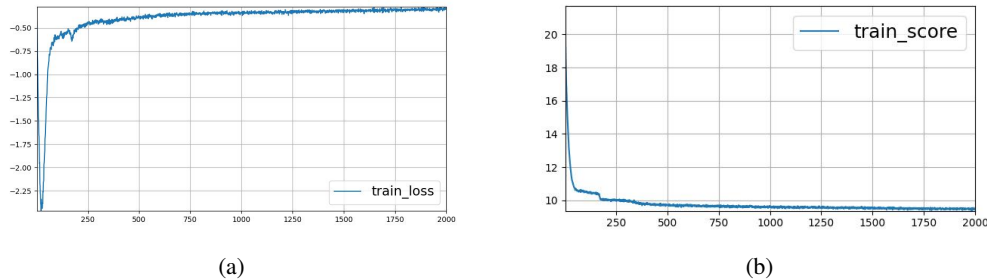

(a)  (b)

Figure 6: The training loss and score curves for R-50-100 instances. The horizontal axis represents epochs, while the vertical axis represents the values of loss or score (i.e. the objective value). (a) Training loss. (b) Training score.

For larger-scale instances with 50 nodes, training is performed on three GPUs. The batch size is adjusted to 25 to accommodate the GPU memory limitations. The learning rate for the Adam optimizer is set to $\alpha = 2 \times 10^{-4}$. Each GPU handles 400 training instances per epoch. The models are uniformly trained for 4000 epochs.

As for the built-in CVRP models, they are trained for 5000 epochs to ensure their effectiveness and convergence.

## 7.6 Effectiveness of Budget Uncertainty Set

The hyperparameter $\Gamma$ can be leveraged to adjust the robustness of the budget uncertainty set. Nevertheless, we conducted supplementary experiments to validate the effectiveness of our approach under general budget uncertainty conditions. Table 6 presents the comparative results for different values. Our method shows highly promising outcomes within a remarkably short time, especially when compared to the leading solver, EGA.

Table 6: Comparison result of different $\Gamma$ on $N = 20$.

| Method | Obj | Time($s$) | Obj | Time($s$) | Obj | Time($s$) |
|---|---|---|---|---|---|---|
| | $\Gamma = \lfloor \frac{C(N,2)}{2} \rfloor$ | | $\Gamma = \lfloor \frac{C(N,2)}{4} \rfloor$ | | $\Gamma = 0$ | |
| EGA | **0.7870** | 55.9 | **0.3175** | 56.7 | **0.0000** | 38.4 |
| ours*128 | 0.7870 | 11.2 | 0.3180 | 10.9 | 0.0005 | 11.3 |
| ours*8 | 0.7945 | **0.8** | 0.3305 | **0.8** | 0.0365 | **0.8** |

## 7.7 Architecture of MatNet

MatNet [30] is a graph attention network model designed for data objects that can be partitioned into two sets of nodes in a bipartite graph. To facilitate the description, we refer to the set of nodes acting as the heads of edges as $\Phi$, and the set of nodes acting as the tails of edges as $\Psi$. The framework diagram of MatNet is depicted in Figure 7. MatNet consists of a stack of $L$ attention layers. Each attention layer comprises two sub-layers: the multi-head mixed-score cross-attention layer and the fully connected feed-forward (FF) layer. Both sub-layers incorporate skip connections and batch normalization. In the FF sub-layer, a hidden layer dimension of 512 and a ReLU activation function are employed. The multi-head mixed-score cross-attention sub-layer calculates attention scores between each node in one set and all nodes in the other set. This mechanism enables the learning of association features between the two sets of nodes.

The update of two node sets $\Phi$ and $\Psi$ is performed in a dual manner, where the roles of queries and keys/values are exchanged, accompanied by the transposition of the matrix. Taking the update of $h_{\phi i}^{l}$ as an example, where $l$ denotes the current encoder layer being updated, and $h_{\phi i}$ represents the embedding representation of node $i$ in the node set $\Phi$. We define the query $q_i$ for each node $i$ in the set $\Phi$ by projecting the embedding $h_{\phi i}^{l-1}$. Simultaneously, we obtain the key $k_j$ and value $v_j$ from the embedding $h_{\psi j}^{l-1}$.

$$q_i = W^Q h_{\phi i}^{l-1}, k_j = W^K h_{\psi j}^{l-1}, v_j = W^V h_{\psi j}^{l-1}, \tag{30}$$

The compatibility $u_{ij}$ is calculated using $q_i$ and $k_j$:

$$u_{ij} = \frac{q_i k_j^T}{\sqrt{d_k}}, \tag{31}$$

where $d_k$ is the dimensions of the attention vector $k$ and $v$.

During the feature update process of the nodes in the attention layer, edge weight information represented by a matrix is added with a multi-layer perceptron (MLP) in addition to the network output of the previous layer.

$$weights = softmax((w_{s2}(ReLu(w_{s1}scores^T + b_{s1})) + b_{s2})^T), \tag{32}$$

where $scores_{ij} = u_{ij} \| d_{ij}$, and $d_{ij}$ denotes the corresponding distance matrix element.

Next, update the corresponding node embedding.

$$h_{\phi i}^{l-1'} = \sum_j weights_{ij} v_j, \tag{33}$$

Then, proceed through the multi-head sub-layer and the feed-forward sub-layer sequentially to obtain the updated embedding representation for the next layer.

$$MHA_{\phi i}^l(h_{\phi i}^{l-1}, \{h_{\psi 1}^{l-1}, .., h_{\psi N}^{l-1}\}) = \sum_{m=1}^{M} W_m^O h_{\phi im}^{l-1'}, \tag{34}$$

$$\hat{h}_{\phi i}^l = BN_{\phi i}^l(h_{\phi i}^{l-1} + MHA_{\phi i}^l(h_{\phi i}^{l-1}, \{h_{\psi 1}^{l-1}..h_{\psi N}^{l-1}\})), \tag{35}$$

$$h_{\phi i}^l = BN_{\phi i}^l(\hat{h}_{\phi i}^l + FF_{\phi i}^l(\hat{h}_{\phi i}^l)), \tag{36}$$

As depicted in Figure 7, the final node feature embedding representation for the two node sets can be obtained y repeating the above process $L$ times.

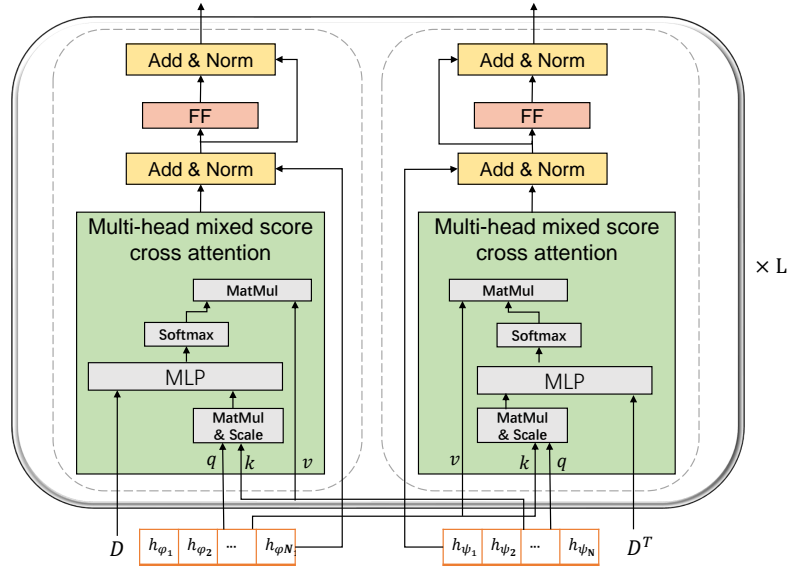

Figure 7: The schematic of MatNet for a single bipartite graph. $D$ is the corresponding matrix representation of the bipartite graph.

